# Blind Separation of Filtered Sources Using State-Space Approach

**Liqing Zhang**\* **and Andrzej Cichocki**[†]
Laboratory for Open Information Systems,
Brain Science Institute, RIKEN
Saitama 351-0198, Wako shi, JAPAN
Email: {zha, cia}@open.brain.riken.go.jp

## Abstract

In this paper we present a novel approach to multichannel blind separation/generalized deconvolution, assuming that both mixing and demixing models are described by stable linear state-space systems. We decompose the blind separation problem into two process: separation and state estimation. Based on the minimization of Kullback-Leibler Divergence, we develop a novel learning algorithm to train the matrices in the output equation. To estimate the state of the demixing model, we introduce a new concept, called hidden innovation, to numerically implement the Kalman filter. Computer simulations are given to show the validity and high effectiveness of the state-space approach.

## 1 Introduction

The field of blind separation and deconvolution has grown dramatically during recent years due to its similarity to the separation feature in human brain, as well as its rapidly growing applications in various fields, such as telecommunication systems, image enhancement and biomedical signal processing. The blind source separation problem is to recover independent sources from sensor outputs without assuming any priori knowledge of the original signals besides certain statistic features. Refer to review papers [1] and [5] for the current state of theory and methods in the field.

Although there exist a number of models and methods, such as the infomax, natural gradient approach and equivariant adaptive algorithms, for separating blindly independent sources, there still are several challenges in generalizing mixture to dy-

[†]On leave from Warsaw University of Technology, Poland

namic and nonlinear systems, as well as in developing more rigorous and effective algorithms with general convergence.[1-9], [11-13]

The state-space description of systems is a new model for blind separation and deconvolution[9,12]. There are several reasons why we use linear state-space systems as blind deconvolution models. Although transfer function models are equivalent to the state-space ones, it is difficult to exploit any common features that may be present in the real dynamic systems. The main advantage of the state space description for blind deconvolution is that it not only gives the internal description of a system, but there are various equivalent types of state-space realizations for a system, such as balanced realization and observable canonical forms. In particular it is known how to parameterize some specific classes of models which are of interest in applications. Also it is much easy to tackle the stability problem of state-space systems using the Kalman Filter. Moreover, the state-space model enables much more general description than standard finite impulse response (FIR) convolutive filtering. All known filtering (dynamic) models, like AR, MA, ARMA, ARMAX and Gamma filterings, could also be considered as special cases of flexible state-space models.

## 2 Formulation of Problem

Assume that the source signals are a stationary zero-mean i.i.d processes and mutually statistically independent. Let $s(t) = (s_1(t), \cdots, s_n(t))$ be an unknown vector of independent i.i.d. sources. Suppose that the mixing model is described by a stable linear state discrete-time system

$$\overline{x}(k+1) = \overline{A}\overline{x}(k) + \overline{B}s(k) + \overline{L}\xi_P(k), \tag{1}$$

$$u(k) = \overline{C}\overline{x}(k) + \overline{D}s(k) + \theta(k), \tag{2}$$

where $\overline{x} \in \mathbf{R}^r$ is the state vector of system, $s(k) \in \mathbf{R}^n$ is the vector of source signals and $u(k) \in \mathbf{R}^m$ is the vector of sensor signals. $\overline{A}$, $\overline{B}$, $\overline{C}$ and $\overline{D}$ are the mixing matrices of the state space model with consistent dimensions. $\xi_P(k)$ is the process noise and $\theta(k)$ is sensor noise of the mixing system. If we ignore the noise terms in the mixing model, its transfer function matrix is described by a $m \times n$ matrix of the form

$$H(z) = \overline{C}(z\mathbf{I} - \overline{A})^{-1}\overline{B} + \overline{D}, \tag{3}$$

where $z^{-1}$ is a delay operator.

We formulate the blind separation problem as a task to recover original signals from observations $u(t)$ without prior knowledge on the source signals and the state space matrices $[\overline{A}, \overline{B}, \overline{C}, \overline{D}]$ besides certain statistic features of source signals. We propose that the demixing model here is another linear state-space system, which is described as follows, (see Fig. 1)

$$x(k+1) = Ax(k) + Bu(k) + L\xi_R(k), \tag{4}$$

$$y(k) = Cx(k) + Du(k), \tag{5}$$

where the input $u(k)$ of the demixing model is just the output (sensor signals) of the mixing model and the $\xi_R(k)$ is the reference model noise. $A$, $B$, $C$ and $D$ are the demixing matrices of consistent dimensions. In general, the matrices $W = [A, B, C, D, L]$ are parameters to be determined in learning process.

For simplicity, we do not consider, at this moment, the noise terms both in the mixing and demixing models. The transfer function of the demixing model is $W(z) = C(z\mathbf{I} - A)^{-1}B + D$. The output $y(k)$ is designed to recover the source signals in the following sense

$$y(k) = W(z)H(z)s(k) = P\Lambda(z)s(k), \tag{6}$$

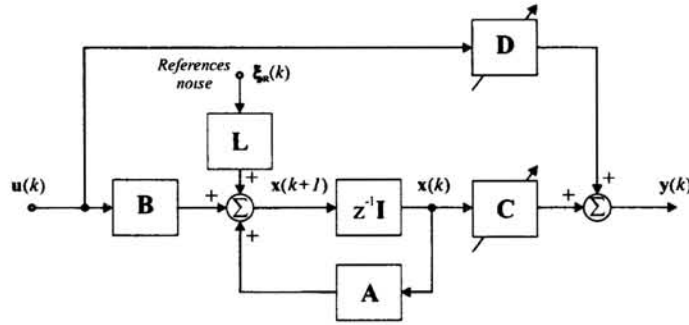

Figure 1: General state-space model for blind deconvolution

where $\boldsymbol{P}$ is any permutation matrix and $\Lambda(z)$ is a diagonal matrix with $\lambda_i z^{-\tau_i}$ in diagonal entry (i,i), here $\lambda_i$ is a nonzero constant and $\tau_i$ is any nonnegative integer. It is easy to see that the linear state space model mixture is an extension of instantaneous mixture. When both the matrices $\overline{\boldsymbol{A}}, \overline{\boldsymbol{B}}, \overline{\boldsymbol{C}}$ in the mixing model and $\boldsymbol{A}, \boldsymbol{B}, \boldsymbol{C}$ in the demixing model are null matrices, the problem is simplified to standard ICA problem [1-8].

The question here is whether exist matrices $[\boldsymbol{A}, \boldsymbol{B}, \boldsymbol{C}, \boldsymbol{D}]$ in the demixing model (4) and (5), such that its transfer function $\boldsymbol{W}(z)$ satisfies (6). It is proven [12] that if the matrix $\overline{\boldsymbol{D}}$ in the mixing model is of full rank, $rank(\overline{\boldsymbol{D}}) = n$, then there exist matrices $[\boldsymbol{A}, \boldsymbol{B}, \boldsymbol{C}, \boldsymbol{D}]$, such that the output signal $\boldsymbol{y}$ of state-space system (4) and (5) recovers the independent source signal $\boldsymbol{s}$ in the sense of (6).

## 3   Learning Algorithm

Assume that $p(\boldsymbol{y}, \boldsymbol{W}), p_i(y_i, \boldsymbol{W})$ are the joint probability density function of $\boldsymbol{y}$ and marginal pdf of $y_i$, $(i = 1, \cdots, n)$ respectively. We employ the mutual information of the output signals, which measures the mutual independence of the output signals $y_i(k)$, as a risk function [1,2]

$$l(\boldsymbol{W}) = -H(\boldsymbol{y}, \boldsymbol{W}) + \sum_{i=1}^{n} H(y_i, \boldsymbol{W}),  \qquad (7)$$

where

$$H(\boldsymbol{y}, \boldsymbol{W}) = -\int p(\boldsymbol{y}, \boldsymbol{W}) \log p(\boldsymbol{y}, \boldsymbol{W}) dy, \ \ H(y_i, \boldsymbol{W}) = -\int p_i(y_i, \boldsymbol{W}) \log p_i(y_i, \boldsymbol{W}) dy_i.$$

In this paper we do not directly develop learning algorithms to update all parameters $\boldsymbol{W} = [\boldsymbol{A}, \boldsymbol{B}, \boldsymbol{C}, \boldsymbol{D}]$ in demixing model. We separate the blind deconvolution problem into two procedures: separation and state-estimation. In the separation procedure we develop a novel learning algorithm, using a new search direction, to update the matrices $\boldsymbol{C}$ and $\boldsymbol{D}$ in output equation (5). Then we define a hidden innovation of the output and use Kalman filter to estimate the state vector $\boldsymbol{x}(k)$.

For simplicity we suppose that the matrix $\boldsymbol{D}$ in the demixing model (5) is nonsingular $n \times n$ matrix. From the risk function (7), we can obtain a cost function for on line learning

$$l(\boldsymbol{y}, \boldsymbol{W}) = -\frac{1}{2} \log det(\boldsymbol{D}^T \boldsymbol{D})) - \sum_{i=1}^{n} \log p_i(y_i, \boldsymbol{W}),  \qquad (8)$$

where $det(\boldsymbol{D}^T\boldsymbol{D})$ is the determinant of symmetric positive definite matrix $\boldsymbol{D}^T\boldsymbol{D}$. For the gradient of $l$ with respect to $\boldsymbol{W}$, we calculate the total differential $dl$ of $l(\boldsymbol{y},\boldsymbol{W})$ when we takes a differential $d\boldsymbol{W}$ on $\boldsymbol{W}$

$$dl(\boldsymbol{y},\boldsymbol{W}) = l(\boldsymbol{y},\boldsymbol{W}+d\boldsymbol{W}) - l(\boldsymbol{y},\boldsymbol{W}). \tag{9}$$

Following Amari's derivation for natural gradient methods [1-3], we have

$$dl(\boldsymbol{y},\boldsymbol{W}) = -tr(d\boldsymbol{D}\boldsymbol{D}^{-1}) + \boldsymbol{\varphi}^T(\boldsymbol{y})d\boldsymbol{y}, \tag{10}$$

where $tr$ is the trace of a matrix and $\boldsymbol{\varphi}(\mathbf{y})$ is a vector of nonlinear activation functions

$$\varphi_i(\boldsymbol{y}_i) = -\frac{d\log p_i(y_i)}{dy_i} = -\frac{p_i'(y_i)}{p_i(y_i)}. \tag{11}$$

Taking the derivative on equation (5), we have following approximation

$$d\boldsymbol{y} = d\boldsymbol{C}\boldsymbol{x}(k) + d\boldsymbol{D}\boldsymbol{u}(k). \tag{12}$$

On the other hand, from (5), we have

$$\boldsymbol{u}(k) = \boldsymbol{D}^{-1}(\boldsymbol{y}(k) - \boldsymbol{C}\boldsymbol{x}(k)) \tag{13}$$

Substituting (13) into (12), we obtain

$$d\boldsymbol{y} = (d\boldsymbol{C} - d\boldsymbol{D}\boldsymbol{D}^{-1}\boldsymbol{C})\boldsymbol{x} + d\boldsymbol{D}\boldsymbol{D}^{-1}\boldsymbol{y}. \tag{14}$$

In order to improve the computing efficiency of learning algorithms, we introduce a new search direction

$$d\boldsymbol{X}_1 = d\boldsymbol{C} - d\boldsymbol{D}\boldsymbol{D}^{-1}\boldsymbol{C}, \tag{15}$$

$$d\boldsymbol{X}_2 = d\boldsymbol{D}\boldsymbol{D}^{-1}. \tag{16}$$

Then the total differential $dl$ can be expressed by

$$dl = -tr(d\boldsymbol{X}_2) + \boldsymbol{\varphi}^T(\boldsymbol{y})(d\boldsymbol{X}_1\boldsymbol{x} + d\boldsymbol{X}_2\boldsymbol{y}). \tag{17}$$

It is easy to obtain the derivatives of the cost function $l$ with respect to matrices $\boldsymbol{X}_1$ and $\boldsymbol{X}_2$ as

$$\frac{\partial l}{\partial \boldsymbol{X}_1} = \boldsymbol{\varphi}(\boldsymbol{y}(k))\boldsymbol{x}^T(k), \tag{18}$$

$$\frac{\partial l}{\partial \boldsymbol{X}_2} = \boldsymbol{\varphi}(\boldsymbol{y}(k))\boldsymbol{y}^T(k) - \boldsymbol{I}. \tag{19}$$

From (15) and (16), we derive a novel learning algorithm to update matrices $\boldsymbol{C}$ and $\boldsymbol{D}$.

$$\Delta\boldsymbol{C}(k) = \eta\left(-\boldsymbol{\varphi}(\boldsymbol{y}(k))\boldsymbol{x}^T(k) + (\boldsymbol{I} - \boldsymbol{\varphi}(\boldsymbol{y}(k))\boldsymbol{y}^T(k))\boldsymbol{C}(k)\right), \tag{20}$$

$$\Delta\boldsymbol{D}(k) = \eta\left(\boldsymbol{I} - \boldsymbol{\varphi}(\boldsymbol{y}(k))\boldsymbol{y}^T(k)\right)\boldsymbol{D}(k). \tag{21}$$

The equilibrium points of the learning algorithm satisfy the following equations

$$E[\boldsymbol{\varphi}(\boldsymbol{y}(k))\boldsymbol{x}^T(k)] = 0, \tag{22}$$

$$E\left[\boldsymbol{I} - \boldsymbol{\varphi}(\boldsymbol{y}(k))\boldsymbol{y}^T(k)\right] = 0. \tag{23}$$

This means that separated signals $\boldsymbol{y}$ could achieve as mutually independent as possible if the nonlinear activation function $\boldsymbol{\varphi}(\boldsymbol{y})$ are suitably chosen and the state vector $\boldsymbol{x}(k)$ is well estimated. From (20) and (21), we see that the natural gradient learning algorithm [2] is covered as a special case of the learning algorithm when the mixture is simplified to instantaneous case.

The above derived learning algorithm enable to solve the blind separation problem under assumption that state matrices $\boldsymbol{A}$ and $\boldsymbol{B}$ are known or designed appropriately. In the next section instead of adjusting state matrices $\boldsymbol{A}$ and $\boldsymbol{B}$ directly, we propose new approaches how to estimate state vector $\boldsymbol{x}$.

## 4   State Estimator

From output equation (5), it is observed that if we can accurately estimate the state vector $x(k)$ of the system, then we can separate mixed signals using the learning algorithm (20) and (21).

### 4.1   Kalman Filter

The Kalman filter is a useful technique to estimate the state vector in state-space models. The function of the Kalman Filter is to generate on line the state estimate of the state $x(k)$. The Kalman filter dynamics are given as follows

$$x(k+1) = Ax(k) + Bu(k) + Kr(k) + \xi_R(k), \qquad (24)$$

where $K$ is the Kalman filter gain matrix, and $r(k)$ is the innovation or residual vector which measures the error between the measured(or expected) output $y(k)$ and the predicted output $Cx(k) + Du(k)$. There are varieties of algorithms to update the Kalman filter gain matrix $K$ as well as the state $x(k)$, refer to [10] for more details.

However in the blind deconvolution problem there exists no explicit residual $r(k)$ to estimate the state vector $x(k)$ because the expected output $y(t)$ means the unavailable source signals. In order to solve this problem, we present a new concept called hidden innovation to implement the Kalman filter in blind deconvolution case. Since updating matrices $C$ and $D$ will produces an innovation in each learning step, we introduce a hidden innovation as follows

$$r(k) = \Delta y(k) = \Delta Cx(k) + \Delta Du(k), \qquad (25)$$

where $\Delta C = C(k+1) - C(k)$ and $\Delta D = D(k+1) - D(k)$. The hidden innovation presents the adjusting direction of the output of the demixing system and is used to generate an a posteriori state estimate. Once we define the hidden innovation, we can employ the commonly used Kalman filter to estimate the state vector $x(k)$, as well as to update Kalman gain matrix $K$. The updating rule in this paper is described as follows:
(1) Compute the Kalman gain matrix

$$K(k) = P(k)C(k)^T(C(k)P(k)C^T(k) + R(k))^{-1}$$

(2) Update state vector with hidden innovation

$$\hat{x}(k) = x(k) + K(k)r(k)$$

(3) Update the error covariance matrix

$$\hat{P}(k) = (I - K(k)C(k))P(k)$$

(4) evaluate the state vector ahead

$$x_{k+1} = A(k)\hat{x}(k) + B(k)u(k)$$

(5) evaluate the error covariance matrix ahead

$$P(k) = A(k)\hat{P}(k)A(k)^T + Q(k)$$

with the initial condition $P(0) = I$, where $Q(k)$, $R(k)$ are the covariance matrices of the noise vector $\xi_R$ and output measurement noise $n_k$.

The theoretic problems, such as convergence and stability, remain to be elaborated. Simulation experiments show that the proposed algorithm, based on the Kalman filter, can separate the convolved signals well.

## 4.2 Information Back-propagation

Another solution to estimating the state of a system is to propagate backward the mutual information. If we consider the cost function is also a function of the vector $x$, than we have the partial derivative of $l(y, W)$ with respect to $x$

$$\frac{\partial l(y, W)}{\partial x} = C^T \varphi(y). \tag{26}$$

Then we adjust the state vector $x(k)$ according to the following rule

$$\hat{x}(k) = x(k) - \eta C(k)^T \varphi(y(k)). \tag{27}$$

Then the estimated state vector is used as a new state of the system.

## 5 Numerical Implementation

Several numerical simulations have been done to demonstrate the validity and effectiveness of the proposed algorithm. Here we give a typical example

**Example 1.** Consider the following MIMO mixing model

$$u(k) + \sum_{i=1}^{10} A_i u(k-i) = s(k) + \sum_{i=1}^{10} B_i s(k-i) + v(k),$$

where $u, s, v \in \mathbf{R}^3$, and

$$A_2 = \begin{pmatrix} -0.48 & -0.16 & -0.64 \\ -0.16 & -0.48 & -0.24 \\ -0.16 & -0.16 & -0.08 \end{pmatrix}, \quad A_8 = \begin{pmatrix} -0.50 & -0.10 & -0.40 \\ -0.10 & -0.50 & -0.20 \\ -0.10 & -0.10 & -0.10 \end{pmatrix},$$

$$A_{10} = \begin{pmatrix} 0.32 & 0.19 & 0.38 \\ 0.16 & 0.29 & 0.20 \\ 0.08 & 0.08 & 0.10 \end{pmatrix}, \quad B_2 = \begin{pmatrix} 0.42 & 0.21 & 0.14 \\ 0.10 & 0.56 & 0.14 \\ 0.21 & 0.21 & 0.35 \end{pmatrix},$$

$$B_8 = \begin{pmatrix} -0.40 & -0.08 & -0.08 \\ -0.08 & -0.40 & -0.16 \\ -0.08 & -0.08 & -0.56 \end{pmatrix}, \quad B_{10} = \begin{pmatrix} -0.19 & -0.15 & -0.10 \\ -0.11 & -0.27 & -0.12 \\ -0.16 & -0.18 & -0.22 \end{pmatrix},$$

and other matrices are set to the null matrix. The sources $s$ are chosen to be i.i.d signals uniformly distributed in the range (-1,1), and $v$ are the Gaussian noises with zero mean and a covariance matrix $0.1I$. We employ the state space approach to separate mixing signals. The nonlinear activation function is chosen $\varphi(y) = y^3$. The initial value for matrices $A$ and $B$ in the state equation are chosen as in canonical controller form. The initial values for matrix $C$ is set to null matrix or given randomly in the range (-1,1), and $D = I_3$. A large number of simulations show that the state space method can easily recover source signals in the sense of $W(z)H(z) = P\Lambda$. Figure 2 illustrates the coefficients of global transfer function $G(z) = W(z)H(z)$ after 3000 iterations, where the $(i, j)$th sub-figure plots the coefficients of the transfer function $G_{ij}(z) = \sum_{k=0}^{\infty} g_{ijk} z^{-k}$ up to order of 50.

## Footnotes

\*On leave from South China University of Technology, China

## References

[1] S. Amari and A. Cichocki, "Adaptive blind signal processing– neural network approaches", *Proceedings of the IEEE*, 86(10):2026-2048, 1998.

[2] S. Amari, A. Cichocki, and H.H. Yang, "A new learning algorithm for blind signal separation", *Advances in Neural Information Processing Systems 1995* (Boston, MA: MIT Press, 1996), pp. 752–763.

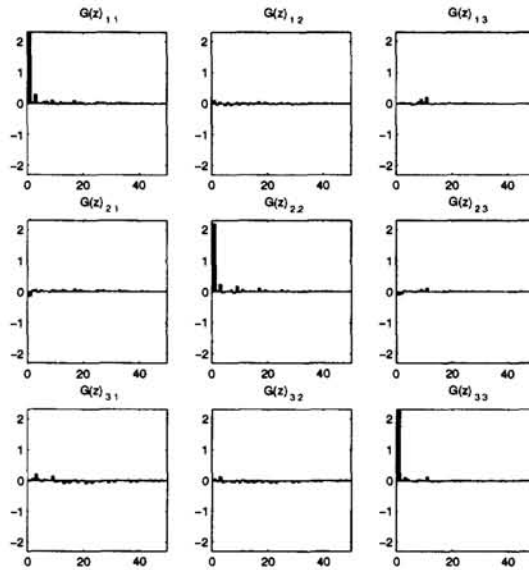

Figure 2: The coefficients of global transfer function after 3000 iterations

[3] S. Amari "Natural gradient works efficiently in learning", *Neural Computation*, Vol.10, pp251-276, 1998.

[4] A. J. Bell and T. J. Sejnowski, " An information-maximization approach to blind separation and blind deconvolution", *Neural Computation*, Vol.7, pp 1129-1159, 1995.

[5] J.-F Cardoso, "Blind signal separation: statistical principles", *Proceedings of the IEEE*, 86(10):2009-2025, 1998.

[6] J.-F. Cardoso and B. Laheld, "Equivariant adaptive source separation," *IEEE Trans. Signal Processing*, vol. SP-43, pp. 3017–3029, Dec. 1996.

[7] A.Cichocki and R. Unbehauen, "Robust neural networks with on-line learning for blind identification and blind separation of sources" IEEE Trans Circuits and Systems I : Fundamentals Theory and Applications, vol 43, No.11, pp. 894-906, Nov. 1996.

[8] P. Comon, "Independent component analysis: a new concept?", *Signal Processing*, vol.36, pp.287–314, 1994.

[9] A. Gharbi and F. Salam, " Algorithm for blind signal separation and recovery in static and dynamics environments", *IEEE Symposium on Circuits and Systems*, Hong Kong, June, 713-716, 1997.

[10] O. L. R. Jacobs, "Introduction to Control Theory", Second Edition, Oxford University Press, 1993.

[11] T. W. Lee, A.J. Bell, and R. Lambert, "Blind separation of delayed and convolved sources", *NIPS 9*, 1997, MIT Press, Cambridge MA, pp758-764.

[12] L. -Q. Zhang and A. Cichocki, "Blind deconvolution/equalization using state-space models", Proc. '98 IEEE Signal Processing Society Workshop on NNSP, pp123-131, Cambridge, 1998.

[13] S. Choi, A. Cichocki and S. Amari, "Blind equalization of simo channels via spatio-temporal anti-Hebbian learning rule", Proc. '98 IEEE Signal Processing Society Workshop on NNSP, pp93-102, Cambridge, 1998.
